# The Robustness-Performance Tradeoff in Markov Decision Processes

**Huan Xu, Shie Mannor**
Department of Electrical and Computer Engineering
McGill University
Montreal, Quebec, Canada, H3A2A7
xuhuan@cim.mcgill.ca
shie@ece.mcgill.ca

## Abstract

Computation of a satisfactory control policy for a Markov decision process when the parameters of the model are not exactly known is a problem encountered in many practical applications. The traditional robust approach is based on a worst-case analysis and may lead to an overly conservative policy. In this paper we consider the tradeoff between nominal performance and the worst case performance over all possible models. Based on parametric linear programming, we propose a method that computes the *whole* set of Pareto efficient policies in the performance-robustness plane when only the reward parameters are subject to uncertainty. In the more general case when the transition probabilities are also subject to error, we show that the strategy with the "optimal" tradeoff might be non-Markovian and hence is in general not tractable.

## 1 Introduction

In many decision problems the parameters of the problem are inherently uncertain. This uncertainty, termed *parameter uncertainty*, can be the result of estimating the parameters from a finite sample or a specification of the parameters that itself includes uncertainty. The standard approach in decision making to circumvent the adverse effect of the parameter uncertainty is to find a solution that performs best under the worst possible parameters. This approach, termed the "robust" approach, has been used in both single stage ([1]) and multi-stage decision problems (e.g., [2]).

In robust optimization problems, it is usually assumed that the constraint parameters are uncertain. By requiring the solution to be feasible to all possible parameters within the uncertainty set, Soyester ([1]) solved the column-wise independent uncertainty case, and Ben-Tal and Nemirovski ([3]) solved the row-wise independent case. In robust MDP problems, there may be two different types of parameter uncertainty, namely, the reward uncertainty and the transition probability uncertainty. Under the assumption that the uncertainty is state-wise independent (an assumption made by all papers to date, to the best of our knowledge), the optimality principle holds and this problem can be decomposed as a series of step by step mini-max problems solved by backward induction ([2, 4, 5]).

The above cited results focus on worst-case analysis. This implies that the vector of *nominal parameters* (the parameters used as an approximation of the true one regardless of the uncertainty) is not treated in a special way and is just an element of the set of feasible parameters. The objective of the worst-case analysis is to eliminate the possibility of disastrous performance. There are several disadvantages to this approach. First, worst-case analysis may lead to an overly conservative solution, i.e., a solution which provides mediocre performance under *all* possible parameters. Second, the desirability of the solution highly depends on the precise modeling of the uncertainty set which is often based on some ad-hoc criterion. Third, it may happen that the nominal parameters are close to

the real parameters, so that the performance of the solution under nominal parameters may provide important information for predicting the performance under the true parameters. Finally, there is a certain tradeoff relationship between the worst-case performance and the nominal performance, that is, if the decision maker insists on maximizing one criterion, the other criterion may decrease dramatically. On the other hand, relaxing both criteria may lead to a well balanced solution with both satisfactory nominal performance and also reasonable robustness to parameter uncertainty.

In this paper we capture the Robustness-Performance (RP) tradeoff explicitly. We use the worst-case behavior of a solution as the function representing its robustness, and formulate the decision problem as an optimization of *both* the robustness criterion and the performance under nominal parameters simultaneously. Here, "simultaneously" is achieved by optimizing the weighted sum of the performance criterion and the robustness criterion. To the best of our knowledge, this is the first attempt to address the overly conservativeness of worst-case analysis in robust MDP.

Instead of optimizing the weighted sum of the robustness and performance for some specific weights, we show how to efficiently find the solutions for all possible weights. We prove that the set of these solutions is in fact equivalent to the set of all Pareto efficient solutions in the robustness-performance space. Therefore, we solve the tradeoff problem *without* choosing a specific tradeoff parameter, and leave the subjective decision of determining the exact tradeoff to the decision maker. Instead of arbitrarily claiming that a certain solution is a good tradeoff, our algorithm computes the whole tradeoff relationship so that the decision maker can choose the most desirable solution according to her preference, which is usually complicated and an explicit form is not available. Our approach thus avoids the tuning of tradeoff parameters, where generally no good a-priori method exists. This is opposed to certain relaxations of the worst-case robust optimization approach like [6] (for single stage only) where some explicit tradeoff parameters have to be chosen. Unlike risk sensitive learning approaches [7, 8, 9] which aim to tune a strategy online, our approach compute a robust strategy off-line without trial and error.

The paper is organized as follows. Section 2 is devoted to the RP tradeoff for Linear Programming. In Section 3 and Section 4 we discuss the RP tradeoff for MDP with uncertain rewards, and uncertain transition probabilities, respectively. In Section 5 we present a computational example. Some concluding remarks are offered in Section 6.

## 2    Parametric linear programming and RP tradeoffs in optimization

In this section, we briefly recall Parametric Linear Programming (PLP) [10, 11, 12], and show how it can be used to find the whole set of Pareto efficient solutions for RP tradeoffs in Linear Programming. This serves as the base for the discussion of RP tradeoffs in MDPs.

### 2.1   Parametric Linear Programming

A Parametric Linear Programming is the following set of infinitely many optimization problems:

$$
\text{For all } \lambda \in [0,1]: \; \begin{aligned} &\text{Minimize:} && \lambda {\mathbf{c}^{(1)}}^{\top}\mathbf{x} + (1-\lambda){\mathbf{c}^{(2)}}^{\top}\mathbf{x} \\ &\text{Subject to:} && A\mathbf{x} = \mathbf{b} \\ &&& \mathbf{x} \geq 0. \end{aligned} \tag{1}
$$

We call ${\mathbf{c}^{(1)}}^{\top}\mathbf{x}$ the first objective, and ${\mathbf{c}^{(2)}}^{\top}\mathbf{x}$ the second objective. We assume that the Linear Program (LP) is feasible and bounded for both objectives. Although there are uncountably many possible $\lambda$, Problem (1) can be solved by a simplex-like algorithm. Here, "solve" means that for each $\lambda$, we find at least one optimal solution. An outline of the PLP algorithm is described in Algorithm 1, which is essentially a tableau simplex algorithm while the entering variable is determined in a specific way. See [10] for a precise description.

**Algorithm 1.**      1. Find a basic feasible optimal solution for $\lambda = 0$. If multiple solutions exist, choose one among those with minimal ${\mathbf{c}^{(1)}}^{\top}\mathbf{x}$.

2. Record current basic feasible solution. Check the reduced cost (i.e., the zero row in the simplex table) of the first objective, denoted as $\bar{c}_{j}^{(1)}$. If none of them is negative, end.

3. Among all columns with negative $\bar{c}_j^{(1)}$, choose the one with largest ratio $|\bar{c}_j^{(1)}/\bar{c}_j^{(2)}|$ as the entering variable.

4. Pivot the base, go to 2.

This algorithm is based on the observation that for any $\lambda$, there exists an optimal basic feasible solution. Hence, by finding a suitable subset of all vertices of the feasible region, we can solve the PLP. Furthermore, we can find this subset by sequentially pivoting among neighboring extreme points like the simplex algorithm does. This algorithm terminates after finitely many iterations. It is also known that the optimal value for PLP is a continuous piecewise linear function of $\lambda$. The theoretical computational cost is exponential, although practically it works well. Such property is shared by all simplex based algorithm. A detailed discussion on PLP can be found in [10, 11, 12].

## 2.2 RP tradeoffs in Linear Programming

Consider the following LP:

$$\text{NOMINAL PROBLEM}: \quad \text{Minimize: } \mathbf{c}^\top \mathbf{x} \tag{2}$$
$$\text{Subject to: } A\mathbf{x} \leq \mathbf{b}$$
$$\text{Here } A \in \mathbb{R}^{n \times m}, \mathbf{x} \in \mathbb{R}^m, \mathbf{b} \in \mathbb{R}^n, \mathbf{c} \in \mathbb{R}^m.$$

Suppose that the constraint matrix $A$ is only a guess of the unknown true parameter $A^r$ which is known to belonging to set $\mathbb{A}$ (we call $\mathbb{A}$ *the uncertainty set*). We assume that $\mathbb{A}$ is constraint-wise independent and polyhedral for each of the constraints. That is, $\mathbb{A} = \prod_{i=1}^n \mathbb{A}_i$, and for each $i$, there exists a matrix $T(i)$ and a vector $\mathbf{v}(i)$ such that $\mathbb{A}_i = \left\{ \mathbf{a}(i)^\top | T(i)\mathbf{a}(i) \leq \mathbf{v}(i) \right\}$.

To quantify how a solution $\mathbf{x}$ behaves with respect to the parameter uncertainty, we define the following criterion to be minimized as its *robustness measure* (more accurately, non-robustness measure).

$$p(\mathbf{x}) \triangleq \sup_{\tilde{A} \in \mathbb{A}} \left\| \left[ \tilde{A}\mathbf{x} - \mathbf{b} \right]^+ \right\|_1$$
$$= \sup_{\tilde{A} \in \mathbb{A}} \sum_{i=1}^n \max \left[ \tilde{\mathbf{a}}(i)^\top \mathbf{x} - b_i, 0 \right] = \sum_{i=1}^n \max \left\{ \left[ \sup_{\tilde{\mathbf{a}}(i):T(i)\tilde{\mathbf{a}}(i)\leq \mathbf{v}(i)} \tilde{\mathbf{a}}(i)^\top \mathbf{x} \right] - b_i, 0 \right\}. \tag{3}$$

Here $[\cdot]^+$ stands for the positive part of a vector, $\tilde{\mathbf{a}}(i)^\top$ is the $i$th row of the matrix $\tilde{A}$, and $b_i$ is the $i$th element of $\mathbf{b}$. In words, the function $p(\mathbf{x})$ is the largest possible sum of constraint violations.

Using the weighted sum of the performance and robustness objective as the minimizing objective, we formulate the explicit tradeoff between robustness and performance as:

$$\text{GENERAL PROBLEM}: \lambda \in [0, 1] \quad \text{Minimize: } \lambda \mathbf{c}^\top \mathbf{x} + (1 - \lambda)p(\mathbf{x})$$
$$\text{Subject to: } A\mathbf{x} \leq \mathbf{b}. \tag{4}$$
$$\text{Here } A \in \mathbb{R}^{n \times m}, \mathbf{x} \in \mathbb{R}^m, \mathbf{b} \in \mathbb{R}^n, \mathbf{c} \in \mathbb{R}^m.$$

By duality theorem, for a given $\mathbf{x}$, $\sup_{\tilde{\mathbf{a}}(i):T(i)\tilde{\mathbf{a}}(i)\leq \mathbf{v}(i)} \tilde{\mathbf{a}}(i)^\top \mathbf{x}$ equals to the optimal value of the following LP on $\mathbf{y}(i)$:

$$\text{Minimize: } \quad \mathbf{v}(i)^\top \mathbf{y}(i)$$
$$\text{Subject to: } \quad T(i)^\top \mathbf{y}(i) = \mathbf{x}$$
$$\mathbf{y}(i) \geq 0.$$

Thus, by adding slack variables, we rewrite GENERAL PROBLEM as the following PLP and solve it using Algorithm 1:

$$\text{GENERAL PROBLEM (PLP)}: \lambda \in [0, 1] \quad \text{Minimize: } \lambda \mathbf{c}^\top \mathbf{x} + (1 - \lambda)\mathbf{1}^\top \mathbf{z}$$
$$\text{Subject to: } A\mathbf{x} \leq \mathbf{b},$$
$$T(i)^\top \mathbf{y}(i) = \mathbf{x},$$
$$\mathbf{v}(i)^\top \mathbf{y}(i) - b_i \leq z_i, \tag{5}$$
$$\mathbf{z} \geq \mathbf{0},$$
$$\mathbf{y}(i) \geq \mathbf{0}; i = 1, 2, \cdots, n.$$

Here, $\mathbf{1}$ stands for a vector of ones of length $n$, $z_i$ is the $i$th element of $\mathbf{z}$, and $\mathbf{x}, \mathbf{y}(i), \mathbf{z}$ are the optimization variables.

# 3 The robustness-performance tradeoff for MDPs with uncertain rewards

A (finite) MDP is defined as a 5-tuple $< T, S, A_s, p(\cdot|s, a), r(s, a) >$ where: $T$ is the (possibly infinite) set of decision stages; $S$ is the state set; $A_s$ is the action set of state $s$; $p(\cdot|s, a)$ is the transition probability; and $r(s, a)$ is the expected reward of state $s$ with action $a \in A_s$. We use $\mathbf{r}$ to denote the vector combining the reward for all state-action pairs and $\mathbf{r}_s$ to denote the vector combining all reward of state $s$. Thus, $r(s, a) = r_s(a)$. Both $S$ and $A_s$ are assumed finite. Both $p$ and $\mathbf{r}$ are time invariant.

In this section, we consider the case where $\mathbf{r}$ is not known exactly. More specifically, we have a nominal parameter $\bar{r}(s, a)$ which is believed to be a reasonably good guess of the true reward. The reward $\mathbf{r}$ is known to belong to a bounded set $\mathfrak{R}$. We further assume that the uncertainty set $\mathfrak{R}$ is state-wise independent and a polytope for each state. That is, $\mathfrak{R} = \prod_{s \in S} \mathfrak{R}_s$, and for each $s \in S$, there exists a matrix $C_s$ and a vector $\mathbf{d}_s$ such that $\mathfrak{R}_s = \{\mathbf{r}_s | C_s \mathbf{r}_s \geq \mathbf{d}_s\}$. We assume that for different visits of one state, the realization of the reward need not be identical and may take different values within the uncertainty set. The set of admissible control policies for the decision maker is the set of randomized history dependent policies, which we denote by $\Pi^{HR}$.

In the following three subsections we discuss different standard reward criteria: cumulative reward with a finite horizon, discounted reward with infinite horizon, and limiting average reward with infinite horizon under a unichain assumption.

## 3.1 Finite horizon case

In the finite horizon case ($T = \{1, \cdots, N\}$), we assume without loss of generality that each state belongs to only one stage, which is equivalent to the assumption of non-stationary reward realization, and use $S_i$ to denote the set of states at the $i^{th}$ stage. We also assume that the first stage consists of only one state $s_1$, and that there are no terminal rewards. We define the following two functions as the performance measure and the robustness measure of a policy $\pi \in \Pi^{HR}$:

$$P(\pi) \triangleq \mathbb{E}_\pi \{ \sum_{i=1}^{N-1} \bar{r}(s_i, a_i) \},$$

$$\tag{6}$$

$$R(\pi) \triangleq \min_{\mathbf{r} \in \mathfrak{R}} \mathbb{E}_\pi \{ \sum_{i=1}^{N-1} r(s_i, a_i) \}.$$

The minimum is attainable, since $\mathfrak{R}$ is compact and the total expected reward is a continuous function of $\mathbf{r}$. We say that a strategy $\pi$ is *Pareto efficient* if it obtains the maximum of $P(\pi)$ among all strategies that have a certain value of $R(\pi)$. The following result is straightforward; the proof can be found in the full version of the paper.

**Proposition 1.** *1. If $\pi^*$ is a Pareto efficient strategy, then there exists a $\lambda \in [0, 1]$ such that $\pi^* \in \arg\max_{\pi \in \Pi^{HR}} \{\lambda P(\pi) + (1 - \lambda) R(\pi)\}$.*

*2. If $\pi^* \in \arg\max_{\pi \in \Pi^{HR}} \{\lambda P(\pi) + (1 - \lambda) R(\pi)\}$ for some $\lambda \in (0, 1)$. Then $\pi^*$ is a Pareto efficient strategy.*

For $0 \leq t \leq N$, $s \in S_t$, and $\lambda \in [0, 1]$ define:

$$P_t(\pi, s) \triangleq \mathbb{E}_\pi \left\{ \sum_{i=t}^{N-1} \bar{r}(s_i, a_i) | s_t = s \right\}$$

$$R_t(\pi, s) \triangleq \min_{\mathbf{r} \in \mathfrak{R}} \mathbb{E}_\pi \left\{ \sum_{i=t}^{N-1} r(s_i, a_i) | s_t = s \right\} \tag{7}$$

$$c_t^\lambda(s) \triangleq \max_{\pi \in \Pi^{HR}} \{\lambda P_t(\pi, s) + (1 - \lambda) R_t(\pi, s)\}.$$

We set $P_N \equiv R_N \equiv c_N \equiv 0$, and note that $c_1^\lambda(s_1)$ is the optimal RP tradeoff with weight $\lambda$. The following theorem shows that the principle of optimality holds for $c$. The proof is omitted since it follows similarly to standard backward induction in finite horizon robust decision problems.

**Theorem 1.** *For $s \in S_t$, $t < N$, let $\Delta_s$ be the probability simplex on $A_s$, then*

$$c_t^\lambda(s) = \max_{\mathbf{q} \in \Delta_s} \left\{ \min_{\mathbf{r}_s \in \mathfrak{R}_s} \left[ \lambda \sum_{a \in A_s} \bar{r}(s,a)q(a) + (1-\lambda) \sum_{a \in A_s} r(s,a)q(a) \right] + \right.$$

$$\left. \sum_{s' \in S_{t+1}} \sum_{a \in A_s} p(s'|s,a)q(a)c_{t+1}^\lambda(s') \right\}.$$

We now consider the maximin problem in each state and show how to find the solutions for all $\lambda$ in one pass. We also prove that $c_t^\lambda(s)$ is piecewise linear in $\lambda$. Let $S_{t+1} = \{s^1, \cdots, s^k\}$. Assume for all $j \in \{1, \cdots, k\}$, $c_{t+1}^\lambda(s^j)$ are continuous piece-wise linear functions. Thus, we can divide $[0,1]$ into finite (say $n$) intervals $[0, \lambda_1], \cdots [\lambda_{n-1}, 1]$ such that in each interval, all $c_{t+1}$ functions are linear. That is, there exist constants $l_i^j$ and $m_i^j$ such that $c_{t+1}^\lambda(s^j) = l_i^j \lambda + m_i^j$, for $\lambda \in [\lambda_{i-1}, \lambda_i]$. By the duality theorem, we have that $c_t^\lambda(s)$ equals to the optimal value of the following LP on $\mathbf{y}$ and $\mathbf{q}$.

$$\text{Maximize: } (1-\lambda)\mathbf{d}_s^\top \mathbf{y} + \lambda \bar{\mathbf{r}}_s^\top \mathbf{q} + \sum_{j=1}^k \sum_{a \in A_s} p(s^j|s,a)q(a)c_{t+1}^\lambda(s^j)$$

$$\text{Subject to: } C_s^\top \mathbf{y} = \mathbf{q},$$
$$\mathbf{1}^\top \mathbf{q} = 1,$$
$$\mathbf{q}, \mathbf{y} \geq 0.$$

(8)

Observe that the feasible set is the same for all $\lambda$. Substituting $c_{t+1}^\lambda(s^j)$ and rearranging, it follows that for $\lambda \in [\lambda_{i-1}, \lambda_i]$ the objective function equals to

$$(1-\lambda)\left\{ \sum_{a \in A_s} \left[ \sum_{j=1}^k p(s^j|s,a)m_i^j \right] q(a) + \mathbf{d}_s^\top \mathbf{y} \right\}$$

$$+\lambda \left\{ \sum_{a \in A_s} \left[ \bar{r}(s,a) + \sum_{j=1}^k p(s^j|s,a)(l_i^j + m_i^j) \right] q(a) \right\}.$$

Thus, for $\lambda \in [\lambda_{i-1}, \lambda_i]$, from the optimal solution for $\lambda_{i-1}$, we can solve for all $\lambda$ using Algorithm 1. Furthermore, we need not to re-initiate for each interval, since the optimal solution for the end of $i^{th}$ interval is also the optimal solution for the begin of the next interval. It is obvious that the resulting $c_t^\lambda(s)$ is also continuous, piecewise linear. Thus, since $c_N = 0$, the assumption of continuous and piecewise linear value functions holds by backward induction.

## 3.2 Discounted reward infinite horizon case

In this section we address the RP tradeoff for infinite horizon MDPs with a discounted reward criterion. For a fixed $\lambda$, the problem is equivalent to a zero-sum game, with the decision maker trying to maximize the weighted sum and *Nature* trying to minimize it by selecting an adversarial reward realization. A well known result in discounted zero-sum stochastic games states that, even if non-stationary policies are admissible, a Nash equilibrium in which both players choose a stationary policy exists; see Proposition 7.3 in [13].

Given an initial state distribution $\alpha(s)$, it is also a known result [14] that there exists a one-to-one correspondence relationship between the state-action frequencies $\sum_{i=1}^\infty \gamma^{i-1}\mathbb{E}(\mathbf{1}_{s_i=s, a_i=a})$ for stationary strategies and vectors belonging to the following polytope $\mathcal{X}$:

$$\sum_{a \in A_{s'}} x(s',a) - \sum_{s \in S} \sum_{a \in A_s} \gamma p(s'|s,a)x(s,a) = \alpha(s'), \quad x(s,a) \geq 0, \ \forall s, \forall a \in A_s.$$

(9)

Since it suffices to consider a stationary policy for Nature, the tradeoff problem becomes:

$$\text{Maximize: } \inf_{\mathbf{r} \in \mathfrak{R}} \sum_{s \in S} \sum_{a \in A_s} [\lambda \bar{r}(s,a)x(s,a) + (1-\lambda)r(s,a)x(s,a)]$$

$$\text{Subject to: } \mathbf{x} \in \mathcal{X}.$$

(10)

By duality of LP, Equation (10) could be rewritten as the following PLP and solved by Algorithm 1.

$$\text{Maximize:}\lambda \sum_{s \in S} \sum_{a \in A_s} \bar{r}(s,a)x(s,a) + (1-\lambda)\sum_{s \in S}\left[\mathbf{d}_s^\top \mathbf{y}_s\right]$$

$$\text{Subject to: } \sum_{a \in A_{s'}} x(s',a) - \sum_{s \in S}\sum_{a \in A_s} \gamma p(s'|s,a)x(s,a) = \alpha(s'), \quad \forall s',$$

$$x(s,a) \geq 0, \quad \forall s, \forall a,$$

$$C_s^\top \mathbf{y}_s = \mathbf{x}_s \quad \forall s,$$

$$\mathbf{y}_s \geq \mathbf{0}, \quad \forall s. \tag{11}$$

### 3.3 Limiting average reward case (unichain)

In the unichain case, the set of limiting average state-action frequency vectors (that is, all limit points of sequences $\left\{\frac{1}{T}\sum_{n=1}^T \mathbb{E}_\pi[\mathbf{1}_{s_n=s,a_n=a}]\right\}$, for $\pi \in \Pi^{HR}$) is the following polytope $\mathcal{X}$:

$$\sum_{a \in A_{s'}} x(s',a) - \sum_{s \in S}\sum_{a \in A_s} p(s'|s,a)x(s,a) = 0, \forall s' \in S,$$

$$\sum_{s \in S}\sum_{a \in A_s} x(s,a) = 1,$$

$$x(s,a) \geq 0, \forall s, \forall a \in A_s. \tag{12}$$

As before, there exists an optimal maximin stationary policy. By a similar argument as for the discounted case, the tradeoff problem can be converted to the following PLP:

$$\text{Maximize:}\lambda \sum_{s \in S}\sum_{a \in A_s} \bar{r}(s,a)x(s,a) + (1-\lambda)\sum_{s \in S}\left[\mathbf{d}_s^\top \mathbf{y}_s\right]$$

$$\text{Subject to: } \sum_{a \in A_{s'}} x(s',a) - \sum_{s \in S}\sum_{a \in A_s} p(s'|s,a)x(s,a) = 0, \quad \forall s',$$

$$\sum_{s \in S}\sum_{a \in A_s} x(s,a) = 1,$$

$$C_s^\top \mathbf{y}_s = \mathbf{x}_s, \quad \forall s,$$

$$\mathbf{y}_s \geq \mathbf{0}, \quad \forall s,$$

$$x(s,a) \geq 0, \quad \forall s, \forall a. \tag{13}$$

## 4 The RP tradeoff in MDPs with uncertain transition probabilities

In this section we provide a counterexample which demonstrates that the weighted sum criterion in the most general case, i.e., the uncertain transition probability case, may lead to non-Markovian optimal policies.

In the finite horizon MDP shown in the Figure 1, $S = \{s1, s2, s3, s4, s5, t1, t2, t3, t4\}$; $A_{s1} = \{a(1,1)\}$; $A_{s2} = \{a(2,1)\}$; $A_{s3} = \{a(3,1)\}$; $A_{s4} = \{a(4,1)\}$ and $A_{s5} = \{a(5,1), a(5,2)\}$. Rewards are only available at the final stage, and are perfectly known. The nominal transition probabilities are $\bar{p}(s2|s1,a(1,1)) = 0.5$, $\bar{p}(s4|s2,a(2,1)) = 1$, and $\bar{p}(t3|s5,a(5,2)) = 1$. The set of possible realization is $p(s2|s1,a(1,1)) \in \{0.5\}$, $p(s4|s2,a(2,1)) \in [0,1]$, and $p(t3|s5,a(5,2)) \in [0,1]$. Observe that the worst parameter realization is $p(s4|s2,a(2,1)) = p(t3|s5,a(5,2)) = 0$. We look for the strategy that maximizes the sum of the nominal reward and the worst-reward (i.e., $\lambda = 0.5$). Since multiple actions only exist in state $s5$, a strategy is determined by the action chosen on $s5$. Let the probability of choosing action $a(5,1)$ and $a(5,2)$ be $p$ and $1-p$, respectively.

Consider the history "$s1 \rightarrow s2$". In this case, with the nominal transition probability, this trajectory will reach $t1$ with a reward of 10, regardless of the choice of $p$. The worst transition is that action $a(2,1)$ leads to $s5$ and action $a(5,2)$ leads to $t4$, hence the expected reward is $5p + 4(1-p)$. Therefore the optimal $p$ equals to 1, i.e., the optimal action is to choose $a(5,1)$ deterministically.

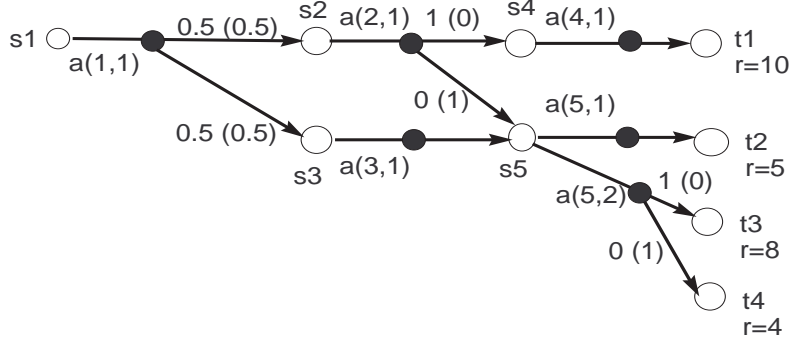

Figure 1: Example of non-Markovian best strategy

Consider the history "$s1 \rightarrow s3$". In this case, the nominal reward is $5p + 8(1 - p)$, and the worst case reward is $5p + 4(1 - p)$. Thus $p = 0$ optimize the weighted sum, i.e., the optimal strategy is to choose $a(5, 2)$.

The unique optimal strategy for this example is thus non-Markovian. This non-Markovian property implies a possibility that past actions affect the choice of future actions, and hence could render the problem intractable. The optimal strategy is non-Markovian because we are taking expectation over two different probability measures, hence the smoothing property of conditional expectation cannot be used in finding the optimal strategy.

## 5 A computational example

We apply our algorithm to a $T$-stage machine maintenance problem. Let $S \triangleq \{1, \cdots, n\}$ denote the state space for each stage. In state $h$, the decision maker can choose either to replace the machine which will lead to state $1$ deterministically, or to continue running, which with probability $p$ will lead to state $h + 1$. If the machine is in state $n$, then the decision maker has to replace it. The replacing cost is perfectly known to be $c_r$, and the nominal running cost in state $h$ is $c_h$. We assume that the realization of the running cost lies in the interval $[c_h - \delta_h, c_h + \delta_h]$. We set $c_h = \sqrt{h} - 1$ and $\delta_h = 2h/n$. The objective is to minimize the total cost, in a risk-averse attitude. Figure 2(a) shows the tradeoff of this MDP.

For each solution found, we sample the reward 300 times according to a uniform distribution. We normalize the cost for each simulation, i.e., we divide the cost by the smallest expected nominal cost. Denoting the normalized cost of the $i_{th}$ simulation for strategy $j$ as $s_i(j)$, we use the following function to compare the solutions:

$$v_j(\alpha) = \sqrt[\alpha]{\frac{\sum_{i=1}^{300} |s_i(j)|^\alpha}{300}}.$$

Note that $\alpha = 1$ is the mean of the simulation cost, whereas larger $\alpha$ puts higher penalty on deviation representing a risk-averse decision maker. Figure 2(b) shows that, the solutions that focus on nominal parameters (i.e., $\lambda$ close to 1) achieve good performance for small $\alpha$, but worse performance for large $\alpha$. That is, if the decision maker is risk neutral, then the solutions based on nominal parameters are good. However, these solutions are not robust and are not good choices for risk-averse decision makers. Note that, in this example, the nominal cost *is* the expected cost for each stage, i.e., the parameters are exactly formulated. Even in such case, we see that risk-averse decision makers can benefit from considering the RP tradeoff.

## 6 Concluding remarks

In this paper we proposed a method that directly addresses the robustness versus performance trade-off by treating the robustness as an optimization objective. Based on PLP, for MDPs where only

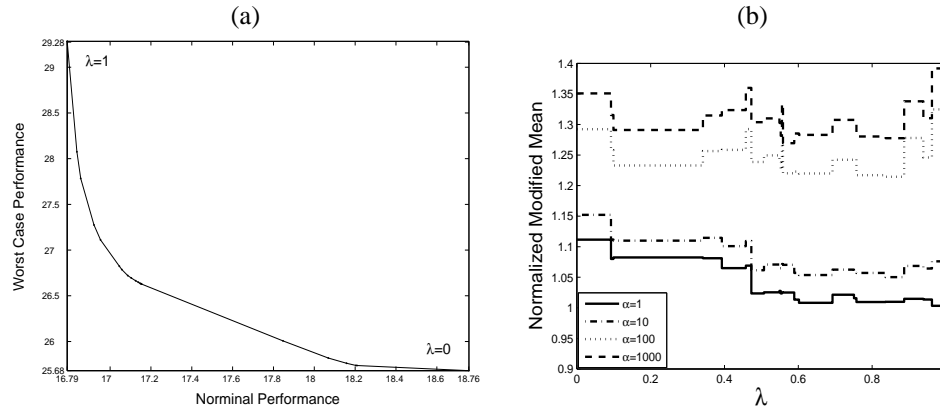

Figure 2: The machine maintenance problem: (a) the PR tradeoff; (b) normalized mean of the simulation for different values of $\alpha$.

rewards are uncertain, we presented an efficient algorithm that computes the whole set of optimal RP tradeoffs for MDPs with finite horizon, infinite horizon discounted reward, and limiting average reward (unichain). For MDPs with uncertain transition probabilities, we showed an example where the solution may be non-Markovian and hence may in general be intractable.

The main advantage of the presented approach is that it addresses robustness directly. This frees the decision maker from the need to make probabilistic assumptions on the problems parameters. It also allows the decision maker to determine the desired robustness-performance tradeoff based on observing the whole curve of possible tradeoffs rather than guessing a single value.

# References

[1] A. L. Soyster. Convex programming with set-inclusive constraints and applications to inexact linear programming. *Oper. Res.*, 1973.

[2] A. Bagnell, A. Ng, and J. Schneider. Solving uncertain markov decision processes. Technical Report CMU-RI-TR-01-25, Carnegie Mellon University, August 2001.

[3] A. Ben-Tal and A. Nemirovski. Robust solutions of uncertain linear programs. *Oper. Res. Lett.*, 25(1):1–13, August 1999.

[4] C. C. White III and H. K. El-Deib. Markov decision process with imprecise transition probabilities. *Oper. Res.*, 42(4):739–748, July 1992.

[5] A. Nilim and L. El Ghaoui. Robust control of markov decision processes with uncertain transition matrices. *Oper. Res.*, 53(5):780–798, September 2005.

[6] D. Bertsimas and M. Sim. The price of robustness. *Oper. Res.*, 52(1):35–53, January 2004.

[7] M. Heger. Consideration of risk in reinforcement learning. In *Proc. 11th International Conference on Machine Learning*, pages 105–111. Morgan Kaufmann, 1994.

[8] R. Neuneier and O. Mihatsch. Risk sensitive reinforcement learning. In *Advances in Neural Information Processing Systems 11*, pages 1031–1037, Cambridge, MA, USA, 1999. MIT Press.

[9] P. Geibel. Reinforcement learning with bounded risk. In *Proc. 18th International Conf. on Machine Learning*, pages 162–169. Morgan Kaufmann, San Francisco, CA, 2001.

[10] D. Bertsimas and J. N. Tsitsiklis. *Introduction to Linear Optimization*. Athena Scientific, 1997.

[11] M. Ehrgott. *Multicriteria Optimization*. Springer-Verlag Berlin Heidelberg, 2000.

[12] K. G. Murty. *Linear Programming*. John Wiley & Sons, 1983.

[13] D. P. Bertsekas and J. N. Tsitsiklis. *Neuro-Dynamic Programming*. Athena Scientific, 1996.

[14] M. L. Puterman. *Markov Decision Processes*. John Wiley & Sons, INC, 1994.
